# Unsmearing Visual Motion: Development of Long-Range Horizontal Intrinsic Connections

Kevin E. Martin        Jonathan A. Marshall

Department of Computer Science, CB 3175, Sitterson Hall
University of North Carolina, Chapel Hill, NC 27599-3175, U.S.A.

## Abstract

Human vision systems integrate information nonlocally, across long spatial ranges. For example, a moving stimulus appears smeared when viewed briefly (30 ms), yet sharp when viewed for a longer exposure (100 ms) (Burr, 1980). This suggests that visual systems combine information along a trajectory that matches the motion of the stimulus. Our self-organizing neural network model shows how developmental exposure to moving stimuli can direct the formation of horizontal trajectory-specific motion integration pathways that unsmear representations of moving stimuli. These results account for Burr's data and can potentially also model other phenomena, such as visual inertia.

## 1  INTRODUCTION

Nonlocal interactions strongly influence the processing of visual motion information and the response characteristics of visual neurons. Examples include: attentional modulation of receptive field shape; modulation of neural response by stimuli beyond the classical receptive field; and neural response to large-field background motion.

In this paper we present a model of the development of nonlocal neural mechanisms for visual motion processing. Our model (Marshall, 1990a, 1991) is based on the long-range excitatory horizontal intrinsic connections (LEHICs) that have been identified in the visual cortex of a variety of animal species (Blasdel, Lund, & Fitzpatrick, 1985; Callaway & Katz, 1990; Gabbott, Martin, & Whitteridge, 1987; Gilbert & Wiesel, 1989; Luhmann, Martínez Millán, & Singer, 1986; Lund, 1987; Michalski, Gerstein, Czarkowska, & Tarnecki, 1983; Mitchison & Crick, 1982; Nelson & Frost, 1985; Rockland & Lund, 1982, 1983; Rockland, Lund, & Humphrey, 1982; Ts'o, Gilbert, & Wiesel, 1986).

## 2  VISUAL UNSMEARING

Human visual systems summate signals over a period of approximately 120 ms in daylight (Burr 1980; Ross & Hogben, 1974). This slow summation reinforces

stationary stimuli but would tend to smear any moving object. Nevertheless, human observers report perceiving both stationary and moving stimuli as sharp (Anderson, Van Essen, & Gallant, 1990; Burr, 1980; Burr, Ross, & Morrone, 1986; Morgan & Benton, 1989; Welch & McKee, 1985). Why do moving objects not appear smeared? Burr (1980) measured perceived smear of moving spots as a function of exposure time. He found that a moving visual spot appears smeared (with a comet-like tail) when it is viewed for a brief exposure (30 ms) yet perfectly sharp when viewed for a longer exposure (100 ms) (Figure 1). The ability to counteract smear at longer exposures suggests that human visual systems combine (or integrate) and sharpen motion information from multiple locations along a specific spatiotemporal *trajectory* that matches the motion of the stimulus (Barlow, 1979, 1981; Burr, 1980; Burr & Ross, 1986) in the domains of direction, velocity, position, and time.

This unsmearing phenomenon also suggests the existence of a memory-like effect, or persistence, which would cause the behavior of processing mechanisms to differ in the early, smeared stages of a spot's motion and in the later, unsmeared stages.

## 3   NETWORK ARCHITECTURE

We built a biologically-modeled self-organizing neural network (SONN) containing long-range excitatory horizontal intrinsic connections (LEHICs) that learns to integrate visual motion information nonlocally. The network laterally propagates predictive moving stimulus information in a trajectory-specific manner to successive image locations where a stimulus is likely to appear. The network uses this propagated information sharpen its representation of visual motion.

### 3.1   LONG-RANGE EXCITATORY HORIZONTAL INTRINSIC CONNECTIONS

The network's LEHICs modeled several characteristics consistent with neurophysiological data:

- They are *highly specific* and anisotropic (Callaway & Katz, 1990).

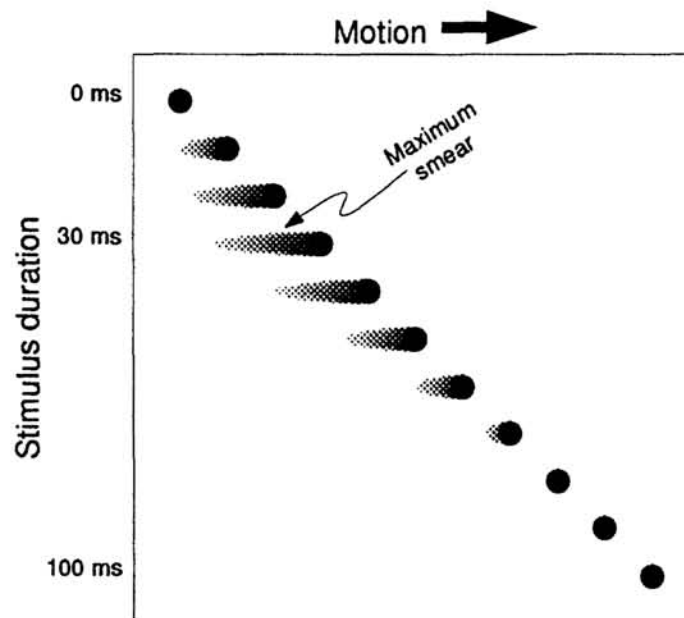

Figure 1: Motion unsmearing. A spot presented for 30 ms appears to have a comet-like tail, but a spot presented for 100 ms appears sharp and unsmeared (Burr, 1980).

- They typically run between neurons with *similar stimulus preferences* (Callaway & Katz, 1990).
- They can run for very *long distances* across the network space (e.g., 10 mm horizontally across cortex) (Luhmann, Martínez Millán, & Singer, 1986).
- They can be *shaped adaptively* through visual experience (Callaway & Katz, 1990; Luhmann, Martínez Millán, & Singer, 1986).
- They may serve to *predictively prime* motion-sensitive neurons (Gabbott, Martin, & Whitteridge, 1987).

Some characteristics of our modeled LEHICs are also consistent with those of the horizontal connections described by Hirsch & Gilbert (1991). For instance, we predicted (Marshall, 1990a) that horizontal excitatory input alone should not cause suprathreshold activation, but horizontal excitatory input should amplify activation when local bottom-up excitation is present. Hirsch & Gilbert (1991) directly observed these characteristics in area 17 pyramidal neurons in the cat.

Since LEHICs are found in early vision processing areas like V1, we hypothesize that similar connections are likely to be found within "higher" cortical areas as well, like areas MT and STS. Our simulated networks may correspond to structures in such higher areas. Although our long-range lateral signals are modeled as being excitatory (Orban, Gulyás, & Vogels, 1987), they are also functionally homologous to long-range trajectory-specific lateral inhibition of neurons tuned to *null*-direction motion (Ganz & Felder, 1984; Marlin, Douglas, & Cynader, 1991; Motter, Steinmetz, Duffy, & Mountcastle, 1987).

LEHICs constitute one possible means by which nonlocal communication can take place in visual cortex. Other means, such as large bottom-up receptive fields, can also cause information to be transmitted nonlocally. However, the main difference between LEHICs and bottom-up receptive fields is that LEHICs provide lateral *feedback* information about the *outcome* of other processing within a given stage. This generates a form of memory, or persistence. Purely bottom-up networks (without LEHICs or other feedback) would perform processing afresh at each step, so that the outcome of processing would be influenced only by the direct, feedforward *inputs* at each step.

## 3.2    RESULTS OF NETWORK DEVELOPMENT

In our model, developmental exposure to moving stimuli guides the formation of motion-integration pathways that unsmear representations of moving stimuli. Our model network is repeatedly exposed to training input sequences of smeared motion patterns through bottom-up excitatory connections. Smear is modeled as an exponential decay and represents the responses of temporally integrating neurons to moving visual stimuli. The network contains a set of initially nonspecific LEHICs with fixed signal transmission latencies. The moving stimuli cause the pattern of weights across the LEHICs to become refined, eventually forming "chains" that correspond to trajectories in the visual environment.

To model unsmearing fully, we would need a 2-D retinotopically organized layer of neurons tuned to different directions of motion and different velocities. Each trajectory in visual space would be represented by a set of like velocity and direction sensitive neurons whose receptive fields are located along the trajectory. These neurons would be connected through a trajectory-specific chain of time-delayed LEHICs. Lateral inhibition between chains would be organized selectively to allow representations of multiple stimuli to be simultaneously active (Marshall, 1990a), thereby letting most trajectory representations operate independently.

Our simulation consists of a 1-D subnetwork of the full 2-D network, with 32 neurons sensitive to a single velocity and direction of motion (Figure 2a). The

lateral inhibitory connections are fixed in a Gaussian distribution, but the LEHIC weights can change according to a modified Hebbian rule (Grossberg, 1982):

$$\frac{d}{dt}z_{ji}^+ = \epsilon\, f(x_i)(-z_{ji}^+ + h(x_j)),$$

where $z_{ji}^+$ represents the weight of the LEHIC from the $j$th neuron to the $i$th neuron, $x_i$ represents the value of the activation level of the $i$th neuron, $\epsilon$ is a slow learning rate, $h(x_j) = \max(0, x_j)^2$ is a faster-than-linear signal function, and $f(x_i) = \max(0, x_i)^2$ is a faster-than-linear sampling function. To model multiple-step trajectories, we used LEHICs with three different signal transmission delays. Initially the LEHICs were all represented, but their weights were zero.

As stimuli move across the receptive fields of the neurons in the network, many neurons are coactive because the network is unable to resolve the smear. By the learning rule, the weights of the LEHICs between these coactive neurons increase. This leads to a profusion of connection weights (Figure 2b), analogous to the "crude clusters" proposed by Callaway and Katz (1990) to describe the early (postnatal days 14-35) structure of horizontal connections in cat area V1.

After sufficient exposure to moving stimuli, the "crude clusters" in our simulation become sharper (Figure 2c) because of the faster-than-linear signal functions. This refinement of the pattern of connection weights into chains might correspond to the later (postnatal day 42+) development of "refined clusters" described by Callaway and Katz (1990).

### 3.3  RESULTS OF NETWORK OPERATION

Before learning begins the network is incapable of unsmearing a stimulus moving across the receptive fields of the neurons (Figure 3a). As the stimulus moves from one position to the next, the pattern of neuron activations is no less smeared than

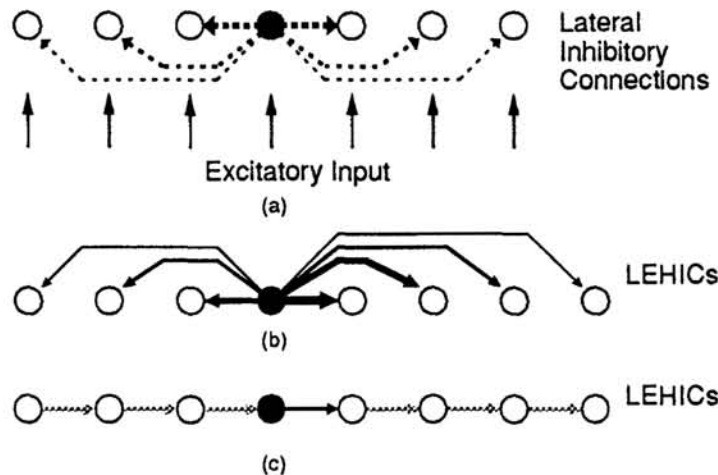

Figure 2: Three phases of modeled development.  (a) Initial.  Lateral excitatory connections were modifiable and had zero weight.  Lateral inhibition was fixed in a Gaussian distribution (thickness of dotted arrows).  The neurons received sequences of smeared rightward-moving excitatory input patterns.  (b) Profusion.  During early development lateral excitatory connections went through a phase of weight profusion.  The output LEHIC weights (thickness of arrows) from one neuron (filled circle) are shown; weights were biased toward rightward motion.  (c) Refinement.  During later development, the pattern of weights settled into sets of regular anisotropic chains; most of the early profuse connections were eliminated.  No external parameters were manipulated during the simulation to induce the initial→profusion→refinement phase transitions.  The simulation contained three different signal transmission latencies, but only one is shown here.

the moving input pattern. No information is built up along the trajectory since the LEHIC weights are still zero.

After training, the network is able to resolve the smear (Figure 3b) in a manner reminiscent of Burr's results (Figure 1). As a stimulus moves, it excites a sequence of neurons whose receptive fields lie along its trajectory. As each neuron receives excitatory input in turn from the moving stimulus, it becomes activated and emits excitatory signals along its trajectory-specific LEHICs. Subsequent neurons along the trajectory then receive both direct stimulus-generated excitation *and* lateral time-delayed excitation. The combination causes these neurons to become even more active; thus activation accumulates along the chain toward an asymptote. The accumulating activation lets neurons farther along the trajectory more effectively suppress (via lateral inhibition) the activation of the neurons carrying the trailing smear. The comet-like tail contracts progressively, and the representation of the

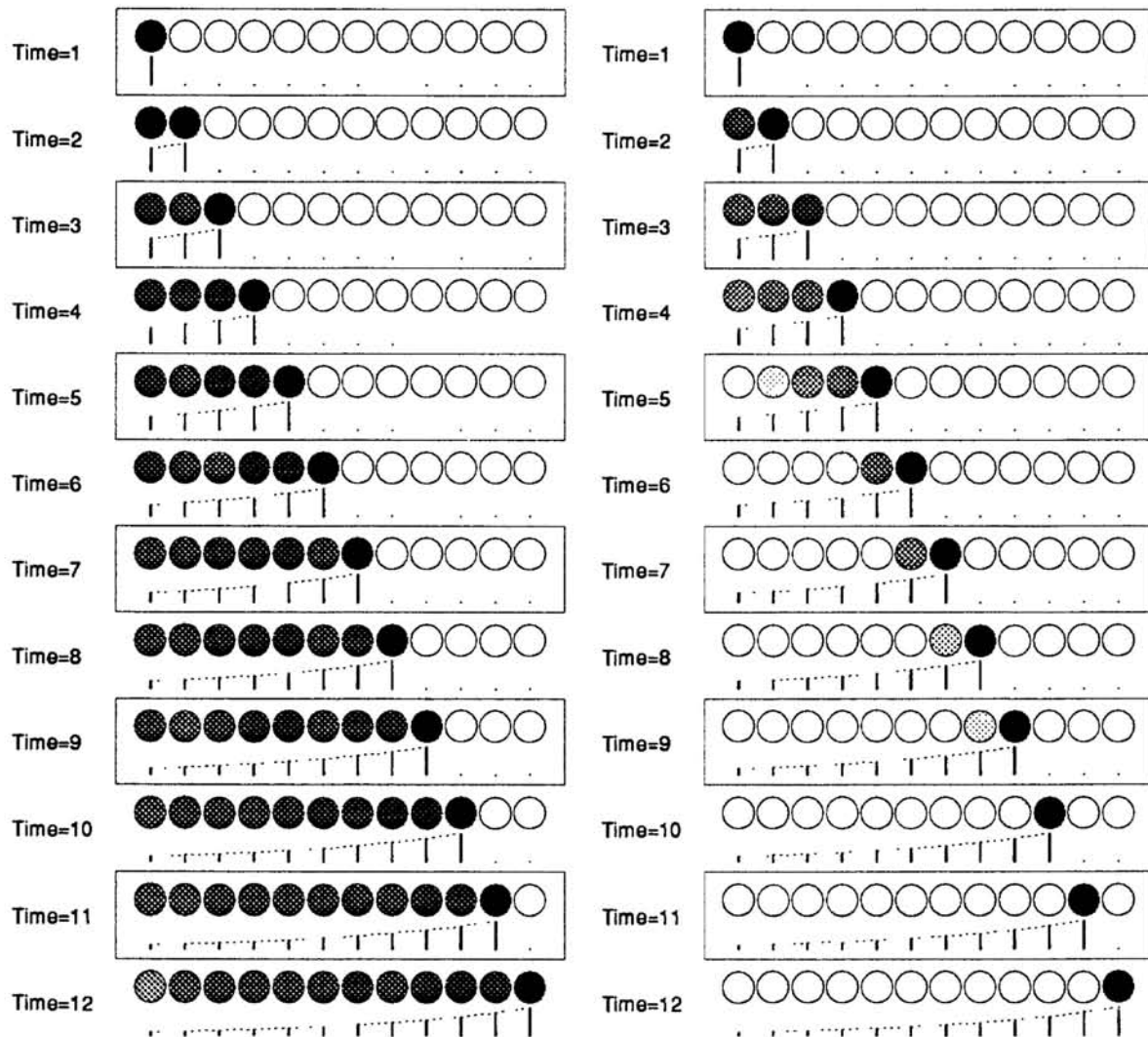

Figure 3: Results of unsmearing simulation. A simulated spot moves rightward for 12 time steps along a 1-D model retina. Smeared input patterns are plotted as vertical lines, and relative output neuron activation patterns are plotted as shading intensity of circles (neurons). (a) Before learning (left) the network is unable to resolve the smear in the input, but (b) after learning (right), the smear is resolved by time step 11. The same test input patterns are used both before and after learning.

moving stimulus becomes increasingly sharp.

Each neuron's activation value $x_i$ changes according to a shunting differential equation (Grossberg, 1982):

$$\frac{d}{dt}x_i = -Ax_i + (B - x_i)E_i - (C + x_i)I_i,$$

where the neuron's total excitatory input $E_i = K_i(1 + L_i)$ combines bottom-up input $K_i$ (the smeared motion) with summed lateral excitation input $L_i = L\sum_j h(x_j)z_{ji}^+$, the neuron's inhibitory input is $I_i = \gamma\sum_j g(x_j)z_{ji}^-$, $h(x_j) = \max(0, x_j)^2$ and $g(x_j) = \max(0, x_j)^3$ are faster-than-linear signal functions, and $A, B, C, L, \beta$, and $\gamma$ are constants.

## 4   CONCLUSIONS AND FUTURE RESEARCH

One might wonder why visual systems allow smear to be represented during the first 30 ms of a stimulus' motion, since simple winner-take-all lateral inhibition could easily eliminate the smear in the representation. Our research leads us to suggest that representing smear lets human visual systems tolerate and even exploit initial uncertainty in local motion measurements. A system with winner-take-all sharpening could not generate a reliable trajectory prediction from an initial inaccurate local motion measurement because the motion can be determined accurately only after multiple measurements along the trajectory are combined (Figure 4a). The inaccurate trajectory predictions of such a network would impair its ability to develop or maintain circuits for combining motion measurements (Marshall, 1990ab). We conclude that motion perception systems need to represent explicitly both initial smear and subsequent unsmearing.

Figure 4a illustrates that when a moving object first appears, the direction in which it will move is uncertain. As the object continues to move, its successor positions become increasingly predictable, in general. The initial smear in the representation is necessary for communicating prior trajectory information to the representations of many possible future trajectory positions.

Faster-than-linear signal functions (Figure 4b) were used so that a neuron would generate little lateral excitation and inhibition when it is uncertain about the presence of the moving stimulus in its receptive field (when a new stimulus appears) and so that a highly active neuron (more certain about the presence of the stimulus in its receptive field) would generate strong lateral excitation and inhibition.

Our results illustrate how visual systems may become able both to propagate motion

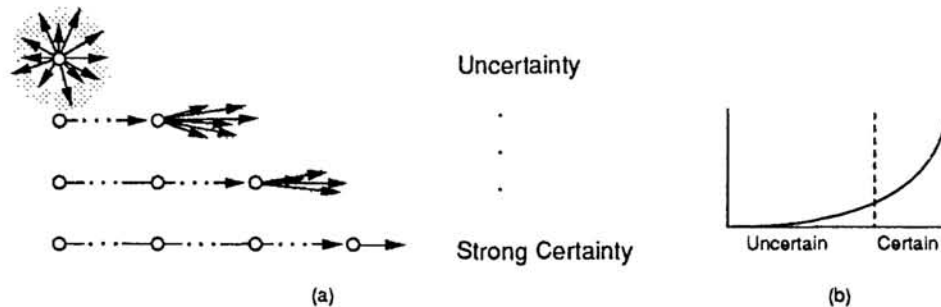

(a)                                                    (b)

Figure 4: Visual motion system uncertainty. (a) When a moving object first appears, the direction in which it will move is uncertain (top row, circular shaded region). As motion proceeds (second, third, and fourth rows), the set of possible stimulus locations becomes increasingly predictable (smaller shaded regions). (b) Faster-than-linear signal functions maintain smear of uncertain data but sharpen more certain data.

information in a trajectory-specific manner and to use the propagated information to unsmear representations of moving objects: (1) Regular anisotropic "chain" patterns of time-delayed horizontal excitatory connections become established through a learning procedure, in response to exposure to ordinary moving visual scenes. (2) Accumulation of propagated motion information along these chains causes a sharpening that unsmears representations of moving visual stimuli.

These results let us model the integration-along-trajectory revealed by Burr's (1980) experiment, within a developmental framework that corresponds to known neurophysiological data; they can potentially also let other nonlocal motion phenomena, such as visual inertia (Anstis & Ramachandran, 1987), be modeled.

## ACKNOWLEDGEMENTS

This work was supported in part by the National Eye Institute (EY09669), by the Office of Naval Research (Cognitive and Neural Sciences, N00014-93-1-0130), and by an Oak Ridge Associated Universities Junior Faculty Enhancement Award.

## REFERENCES

Anderson, C.H., Van Essen, D.C., & Gallant, J.L. (1990). "Blur into Focus." *Nature*, 343, 419–420.

Anstis, S.M. & Ramachandran, V.S. (1987). "Visual Inertia in Apparent Motion." *Vision Research*, 27(5), 755–764.

Barlow, H.B. (1979). "Reconstructing the Visual Image in Space and Time." *Nature*, 279, 189–190.

Barlow, H.B. (1981). "Critical Limiting Factors in the Design of the Eye and Visual Cortex." *Proceedings of the Royal Society of London, Ser. B*, 212, 1–34.

Blasdel, G.G., Lund, J.S., & Fitzpatrick, D. (1985). "Intrinsic Connections of Macaque Striate Cortex: Axonal Projections of Cells Outside Lamina 4C." *Journal of Neuroscience*, 5(12), 3350–3369.

Burr, D. (1980). "Motion Smear." *Nature*, 284, 164–165.

Burr, D. & Ross, J. (1986). "Visual Processing of Motion." *Trends in Neuroscience*, 9(7), 304–307.

Burr, D.C., Ross, J., & Morrone, M.C. (1986). "Seeing Objects in Motion." *Proceedings of the Royal Society of London, Ser. B*, 227, 249–265.

Callaway, E.M. & Katz, L.C. (1990). "Emergence and Refinement of Clustered Horizontal Connections in Cat Striate Cortex." *J. Neurophysiol.*, 10, 1134–1153.

Gabbott, P.L.A., Martin, K.A.C., & Whitteridge, D. (1987). "Connections Between Pyramidal Neurons in Layer 5 of Cat Visual Cortex (Area 17)." *Journal of Comparative Neurology*, 259, 364–381.

Ganz, L. & Felder, R. (1984). "Mechanism of Directional Selectivity is Simple Neurons of the Cat's Visual Cortex Analyzed with Stationary Flash Sequences." *Journal of Neurophysiology*, 51, 294-324.

Gilbert, C.D. & Wiesel, T.N. (1989). "Columnar Specificity of Intrinsic Horizontal and Corticocortical Connections in Cat Visual Cortex." *Journal of Neuroscience*, 9, 2432–2442.

Hirsch, J. & Gilbert, C.D. (1991). "Synaptic Physiology of Horizontal Connections in the Cat's Visual Cortex." *Journal of Neuroscience*, 11, 1800–1809.

Luhmann, H.J., Martínez Millán, L., & Singer, W. (1986). "Development of

Horizontal Intrinsic Connections in Cat Striate Cortex." *Experimental Brain Research*, 63, 443–448.

Lund, J.S. (1987). "Local Circuit Neurons of Macaque Monkey Striate Cortex: I. Neurons of Laminae 4C and 5A." *Journal of Comparative Neurology*, 257, 60–92.

Marlin, S.G., Douglas, R.M., & Cynader, M.S. (1991). "Position-Specific Adaptation in Simple Cell Receptive Fields of the Cat Striate Cortex." *Journal of Neurophysiology*, 66(5), 1769–1784.

Marshall, J.A. (1990a). "Self-Organizing Neural Networks for Perception of Visual Motion." *Neural Networks*, 3, 45–74.

Marshall, J.A. (1990b). "Representation of Uncertainty in Self-Organizing Neural Networks." *Proceedings of the International Neural Network Conference*, Paris, France, July 1990, 809–812.

Marshall, J.A. (1991). "Challenges of Vision Theory: Self-Organization of Neural Mechanisms for Stable Steering of Object-Grouping Data in Visual Motion Perception." Invited Paper, in *Stochastic and Neural Methods in Signal Processing, Image Processing, and Computer Vision*, Su-Shing Chen, Ed., Proceedings of the SPIE 1569, San Diego, CA, July 1991, pp. 200–215.

Michalski, A., Gerstein, G.L., Czarkowska, J., & Tarnecki, R. (1983). "Interactions Between Cat Striate Cortex Neurons." *Experimental Brain Research*, 51, 97–107.

Mitchison, G. & Crick, F. (1982). "Long Axons Within the Striate Cortex: Their Distribution, Orientation, and Patterns of Connection." *Proceedings of the National Academy of Sciences of the U.S.A.*, 79, 3661–3665.

Morgan, M.J. & Benton, S. (1989). "Motion-Deblurring in Human Vision." *Nature*, 340, 385–386.

Motter, B.C., Steinmetz, M.A., Duffy, C.J., & Mountcastle, V.B. (1987). "Functional Properties of Parietal Visual Neurons: Mechanisms of Directionality Along a Single Axis." *Journal of Neuroscience*, 7(1), 154–176.

Nelson, J.I. & Frost, B.J. (1985). "Intracortical Facilitation Among Co-Oriented, Co-Axially Aligned Simple Cells in Cat Striate Cortex." *Experimental Brain Research*, 61, 54–61.

Orban, G.A., Gulyás, B., & Vogels, R. (1987). "Influence of a Moving Textured Background on Direction Selectivity of Cat Striate Neurons." *Journal of Neurophysiology*, 57(6), 1792–1812.

Rockland, K.S. & Lund, J.S. (1982). "Widespread Periodic Intrinsic Connections in the Tree Shrew Visual Cortex." *Science*, 215, 1532–1534.

Rockland, K.S. & Lund, J.S. (1983). "Intrinsic Laminar Lattice Connections in Primate Visual Cortex." *Journal of Comparative Neurology*, 216, 303–318.

Rockland, K.S., Lund, J.S., & Humphrey, A.L. (1982). "Anatomical Banding of Intrinsic Connections in Striate Cortex of Tree Shrews (*Tupaia glis*)." *Journal of Comparative Neurology*, 209, 41–58.

Ross, J. & Hogben, J.H. (1974). *Vision Research*, 14, 1195–1201.

Ts'o, D.Y., Gilbert, C.D., & Wiesel, T.N. (1986). "Relationships Between Horizontal Interactions and Functional Architecture in Cat Striate Cortex as Revealed by Cross-Correlation Analysis." *Journal of Neuroscience*, 6(4), 1160–1170.

Welch, L. & McKee, S.P. (1985). "Colliding Targets: Evidence for Spatial Localization Within the Motion System." *Vision Research*, 25(12), 1901–1910.
